# On Computational Power and the Order-Chaos Phase Transition in Reservoir Computing

**Benjamin Schrauwen**
Electronics and Information Systems Department
Ghent University
B-9000 Ghent, Belgium
benjamin.schrauwen@ugent.be

**Lars Büsing, Robert Legenstein**
Institute for Theoretical Computer Science
Graz University of Technology
A-8010 Graz, Austria
{lars,legi}@igi.tugraz.at

## Abstract

Randomly connected recurrent neural circuits have proven to be very powerful models for online computations when a trained memoryless readout function is appended. Such *Reservoir Computing (RC)* systems are commonly used in two flavors: with analog or binary (spiking) neurons in the recurrent circuits. Previous work showed a fundamental difference between these two incarnations of the RC idea. The performance of a RC system built from binary neurons seems to depend strongly on the network connectivity structure. In networks of analog neurons such dependency has not been observed. In this article we investigate this apparent dichotomy in terms of the in-degree of the circuit nodes. Our analyses based amongst others on the Lyapunov exponent reveal that the phase transition between ordered and chaotic network behavior of binary circuits qualitatively differs from the one in analog circuits. This explains the observed decreased computational performance of binary circuits of high node in-degree. Furthermore, a novel mean-field predictor for computational performance is introduced and shown to accurately predict the numerically obtained results.

## 1   Introduction

In 2001, Jaeger [1] and Maass [2] independently introduced the idea of using a fixed, randomly connected recurrent neural network of simple units as a set of basis filters (operating at the edge-of-stability where the system has fading memory). A memoryless readout is then trained on these basis filters in order to approximate a given time-invariant target operator with fading memory [2]. Jaeger used analog sigmoidal neurons as network units and named the model Echo State Network (ESN). Maass termed the idea Liquid State Machine (LSM) and most of the related literature focuses on networks of spiking neurons or threshold units. Both ESNs and LSMs are special implementations of a concept now generally termed Reservoir Computing (RC) which subsumes the idea of using general dynamical systems (e.g. a network of interacting optical amplifiers [3]) – the so-called reservoirs – in conjunction with trained memoryless readout functions as computational devices. These RC systems have already been used in a broad range of applications (often outperforming other state-of-the-art methods) such as chaotic time-series prediction [4], single digit speech recognition [5], and robot control [6].

Although ESNs and LSMs are based on very similar ideas (and in applications it seems possible to switch between both approaches without loss of performance [7]) an apparent dichotomy exists in the influence of the reservoir's topological structure on its computational performance. The performance of an ESN using analog, rate-based neurons, is e.g. largely independent of the sparsity of the

network [8] or the exact network topology such as small-world or scale-free connectivity graphs[1]. For LSMs, which consist of spiking or binary units, the opposite effect has been observed. For the latter systems, the influence of introducing e.g. small-world or biologically measured lamina-specific cortical interconnection statistics [9] clearly leads to an increase in performance. In the results of [10] it can be observed (although not specifically stated there) that for networks of threshold units with a simple connectivity topology of fixed in-degree per neuron, an increase in performance can be found for decreasing in-degree. None of these effects can be reproduced using ESNs.

In order to systematically study this fundamental difference between binary (spiking) LSMs and analog ESNs, we close the gap between them by introducing in Sec. 2 a class of models termed quantized ESNs. The reservoir of a quantized ESN is defined as a network of discrete units, where the number of admissible states of a single unit is controlled by a parameter called quantization level. LSMs and ESNs can be interpreted as the two limiting cases of quantized ESNs for low and high quantization level respectively. We numerically study the influence of the network topology in terms of the in-degree of the network units on the computational performance of quantized ESNs for different quantization levels. This generalizes and systemizes previous results obtained for binary LSMs and analog ESNs.

In Sec. 3 the empirical results are analyzed by studying the Lyapunov exponent of quantized ESNs, which exhibits a clear relation to the computational performance [11]. It is shown that for ESNs with low quantization level, the chaos-order phase transition is significantly more gradual when the networks are sparsely connected. It is exactly in this transition regime that the computational power of a Reservoir Computing system is found to be optimal [11]. This effect disappears for ESNs with high quantization level. A clear explanation of the influence of the in-degree on the computational performance can be found by investigating the rank measure presented in [11]. This measure characterizes the computational capabilities of a network as a trade-off between the so-called kernel quality and the generalization ability. We show that for highly connected reservoirs with a low quantization level the region of an efficient trade-off implying high performance is narrow. For sparser networks this region is shown to broaden. Consistently for high quantization levels the region is found to be independent of the interconnection degree.

In Sec. 4 we present a novel mean-field predictor for computational power which is able to reproduce the influence of the topology on the quantized ESN model. It is related to the predictor introduced in [10], but it can be calculated for all quantization levels, and can be determined with a significantly reduced computation time. The novel theoretical measure matches the experimental and rank measure findings closely.

## 2   Online Computations with Quantized ESNs

We consider networks of $N$ neurons with the state variable $\mathbf{x}(t) = (x_1(t), \ldots, x_N(t)) \in [-1, +1]^N$ in discrete time $t \in \mathbb{Z}$. All units have an in-degree of $K$, i.e. every unit $i$ receives input from $K$ other randomly chosen units with independently identically distributed (iid.) weights drawn from a normal distribution $\mathcal{N}(0, \sigma^2)$ with zero mean and standard deviation (STD) $\sigma$. The network state is updated according to:

$$x_i(t+1) = (\psi_m \circ g) \left( \sum_{j=1}^{N} w_{ij} x_j(t) + u(t) \right),$$

where $g = \tanh$ is the usual hyperbolic tangent nonlinearity and $u$ denotes the input common to all units. At every time step $t$, the input $u(t)$ is drawn uniformly from $\{-1, 1\}$. The function $\psi_m(\cdot)$ is called quantization function for $m$ bits as it maps from $(-1, 1)$ to its discrete range $\mathcal{S}_m$ of cardinality $2^m$:

$$\psi_m : (-1, 1) \to \mathcal{S}_m, \quad \psi_m(x) := \frac{2\lfloor 2^{m-1}(x+1) \rfloor + 1}{2^m} - 1.$$

Here $\lfloor x \rfloor$ denotes the integer part of $x$. Due to $\psi_m$ the variables $x_i(t)$ are discrete ("quantized") and assume values in $\mathcal{S}_m = \{(2k+1)/2^m - 1 \mid k = 0, \ldots, 2^m - 1\} \subset (-1, 1)$. The network defined above

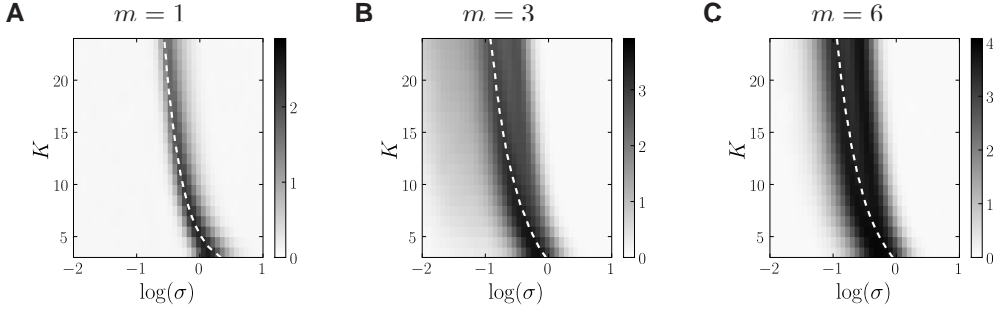

Figure 1: The performance $p_{\exp}(C, \mathrm{PAR}_5)$ for three different quantization levels $m = 1, 3, 6$ is plotted as a function of the network in-degree $K$ and the weight STD $\sigma$. The networks size is $N = 150$, the results have been averaged over 10 circuits $C$, initial conditions and randomly drawn input time series of length $10^4$ time steps. The dashed line represents the numerically determined critical line.

was utilized for online computations on the input stream $u(\cdot)$. We consider in this article tasks where the binary target output at time $t$ depends solely on the $n$ input bits $u(t-\tau-1), \dots, u(t-\tau-n)$ for a given delay parameter $\tau \geq 0$, i.e., it is given by $f_T(u(t-\tau-1), \dots, u(t-\tau-n))$ for a function $f_T \in \{f | f : \{-1, 1\}^n \to \{-1, 1\}\}$. In order to approximate the target output, a linear classifier of the form $\mathrm{sign}(\sum_{i=1}^N \alpha_i x_i(t) + b)$ is applied to the instantaneous network state $\mathbf{x}(t)$. The coefficients $\alpha_i$ and the bias $b$ were trained via a one-shot pseudo-inverse regression method [1]. The RC system consisting of the network and the linear classifier is called a quantized ESN of quantization level $m$ in the remainder of this paper.

We assessed the computational capabilities of a given network based on the numerically determined performance on an example task, which was chosen to be the $\tau$-delayed parity function of $n$ bits $\mathrm{PAR}_{n,\tau}$, i.e. the desired output at time $t$ is $\mathrm{PAR}_{n,\tau}(u, t) = \prod_{i=1}^n u(t-\tau-i)$ for a delay $\tau \geq 0$ and $n \geq 1$. A separate readout classifier is trained for each combination of $n$ and $\tau$, all using the same reservoir. We define $p_{\exp}$ quantifying the performance of a given circuit $C$ on the $\mathrm{PAR}_n$ task as:

$$p_{\exp}(C, \mathrm{PAR}_n) := \sum_{\tau=0}^{\infty} \kappa(C, \mathrm{PAR}_{n,\tau}), \tag{1}$$

where $\kappa(C, \mathrm{PAR}_{n,\tau})$ denotes the performance of circuit $C$ on the $\mathrm{PAR}_{n,\tau}$ task measured in terms of Cohen's kappa coefficient[2]. The performance results for $\mathrm{PAR}_n$ can be considered representative for the general computational capabilities of a circuit $C$ as qualitatively very similar results were obtained for the $\mathrm{AND}_n$ task of $n$ bits and random Boolean functions of $n$ bit (results not shown).

In Fig. 1 the performance $p_{\exp}(C, \mathrm{PAR}_5)$ is shown averaged over 10 circuits $C$ for three different quantization levels $m = 1, 3, 6$. $p_{\exp}(C, \mathrm{PAR}_5)$ is plotted as a function of the network in-degree $K$ and the logarithm[3] of the weight STD $\sigma$. Qualitatively very similar results were obtained for different network graphs with e.g. Poisson or scale-free distributed in-degree with average $K$ (results not shown). A numerical approximation of the critical line, i.e. the order-chaos phase transition, is also shown (dashed line), which was determined by the root of an estimation of the Lyapunov coefficient[4]. The critical line predicts the zone of optimal performance well for $m = 1$, but is less accurate for ESNs with $m = 3, 6$. One can see that for ESNs with low quantization levels ($m = 1, 3$), networks with a small in-degree $K$ reach a significantly better peak performance than those with

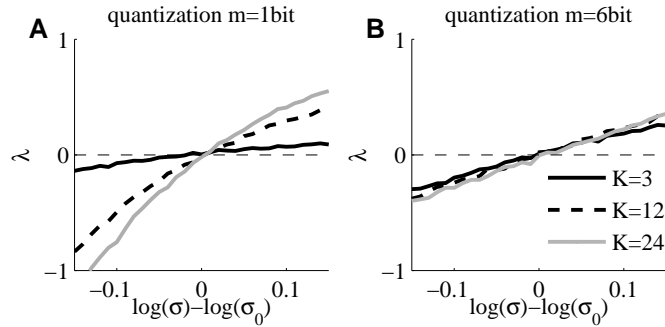

Figure 2: Phase transitions in binary networks ($m = 1$) differ from phase transition in high resolution networks ($m = 6$). An empirical estimate $\lambda$ of the Lyapunov exponent is plotted as a function of the STD of weights $\sigma$ for in-degrees $K = 3$ (solid), $K = 12$ (dashed), and $K = 24$ (gray line). In order to facilitate comparison, the plot for each $K$ is centered around $\log(\sigma_0)$ where $\sigma_0$ is the STD of weights for which $\lambda$ is zero (i.e., $\sigma_0$ is the estimated critical $\sigma$ value for that $K$). The transition sharpens with increasing $K$ for binary reservoirs (**A**), whereas it is virtually independent of $K$ for high resolution reservoirs (**B**).

high in-degree. The effect disappears for a high quantization level ($m = 6$). This phenomenon is consistent with the observation that network connectivity structure is in general an important issue if the reservoir is composed of binary or spiking neurons but less important if analog neurons are employed. Note that for $m = 3, 6$ we see a bifurcation in the zones of optimal performance which is not observed for the limiting cases of ESNs and LSMs.

## 3   Phase Transitions in Binary and High Resolution Networks

Where does the difference between binary and high resolution reservoirs shown in Fig. 1 originate from? It was often hypothesized that high computational power in recurrent networks is located in a parameter regime near the critical line, i.e., near the phase transition between ordered and chaotic behavior (see, e.g., [12] for a review; compare also the performance with the critical line in Fig. 1). Starting from this hypothesis, we investigated whether the network dynamics of binary networks near this transition differs qualitatively from the one of high resolution networks. We estimated the network properties by empirically measuring the Lyapunov exponent $\lambda$ with the same procedure as in the estimation of the critical line in Fig. 1 (see text above). However, we did not only determine the critical line (i.e., the parameter values where the estimated Lyapunov exponent crosses zero), but also considered its values nearby. For a given in-degree $K$, $\lambda$ can then be plotted as a function of the STD of weights $\sigma$ (centered at the critical value $\sigma_0$ of the STD for that $K$). This was done for binary (Fig. 2A) and high resolution networks (Fig. 2B) and for $K = 3, 12$, and $24$. Interestingly, the dependence of $\lambda$ on the STD $\sigma$ near the critical line is qualitatively quite different between the two types of networks. For binary networks the transition becomes much sharper with increasing $K$ which is not the case for high resolution networks.   How can this sharp transition explain the reduced computational performance of binary ESNs with high in-degree $K$? The tasks considered in this article require some limited amount of memory which has to be provided by the reservoir. Hence, the network dynamics has to be located in a regime where memory about recent inputs is available and past input bits do not interfere with that memory. Intuitively, an effect of the sharper phase transition could be stated in the following way. For low $\sigma$ (i.e., in the ordered regime), the memory needed for the task is not provided by the reservoir. As we increase $\sigma$, the memory capacity increases, but older memories interfere with recent ones, making it hard or even impossible to extract the relevant information. This intuition is confirmed by an analysis which was introduced in [11] and which we applied to our setup. We estimated two measures of the reservoir, the so called "kernel-quality" and the "generalization rank", both being the rank of a matrix consisting of certain state vectors of the reservoir. To evaluate the kernel-quality of the reservoir, we randomly drew $N = 150$ input streams $u_1(\cdot), \ldots, u_N(\cdot)$ and computed the rank of the $N \times N$ matrix whose columns were

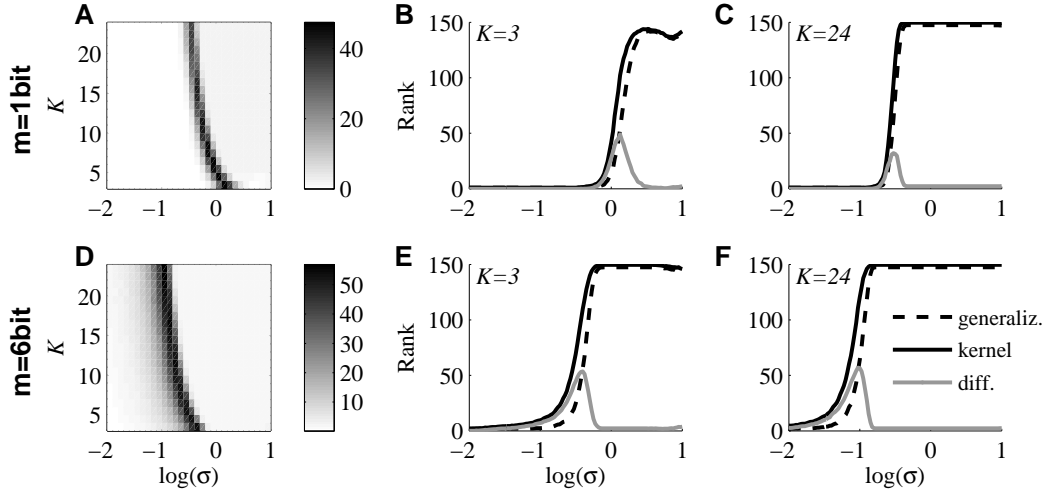

Figure 3: Kernel-quality and generalization rank of quantized ESNs of size $N = 150$. Upper plots are for binary reservoirs ($m = 1$bit), lower plots for high resolution reservoirs ($m = 6$ bit). **A)** The difference between the kernel-quality and the generalization rank as a function of the log STD of weights and the in-degree $K$. **B)** The kernel-quality (solid), the generalization rank (dashed) and the difference between both (gray line) for $K = 3$ as a function of $\log(\sigma)$. **C)** Same as panel B, but for an in-degree of $K = 24$. In comparison to panel B, the transition of both measures is much steeper. **D,E,F)** Same as panels A, B, and C respectively, but for a high resolution reservoir. All plotted values are means over 100 independent runs with randomly drawn networks, initial states, and input streams.

the circuit states resulting from these input streams.[5] Intuitively, this rank measures how well the reservoir represents different input streams. The generalization rank is related to the ability of the reservoir-readout system to generalize from the training data to test data. The generalization rank is evaluated as follows. We randomly drew $N$ input streams $\tilde{u}_1(\cdot), \ldots, \tilde{u}_N(\cdot)$ such that the last three input bits in all these input streams were identical.[6] The generalization rank is then given by the rank of the $N \times N$ matrix whose columns are the circuit states resulting from these input streams. Intuitively, the generalization rank with this input distribution measures how strongly the reservoir state at time $t$ is sensitive to inputs older than three time steps. The rank measures calculated here will thus have predictive power for computations which require memory of the last three time steps (see [11] for a theoretical justification of the measures). In general, a high kernel-quality and a low generalization rank (corresponding to a high ability of the network to generalize) are desirable. Fig. 3A and D show the difference between the two measures as a function of $\log(\sigma)$ and the in-degree $K$ for binary networks and high resolution networks respectively. The plots show that the peak value of this difference is decreasing with $K$ in binary networks, whereas it is independent of $K$ in high resolution reservoirs, reproducing the observations in the plots for the computational performance. A closer look for the binary circuit at $K = 3$ and $K = 24$ is given in Figs. 3B and 3C. When comparing these plots, one sees that the transition of both measures is much steeper for $K = 24$ than for $K = 3$ which leads to a smaller difference between the measures. We interpret this finding in the following way. For $K = 24$, the reservoir increases its separation power very fast as $\log(\sigma)$ increases. However the separation of past input differences increases likewise and thus early input differences cannot be distinguished from late ones. This reduces the computational power of binary ESN with large $K$ on such tasks. In comparison, the corresponding plots for high resolution reservoirs (Figs. 3E and 3F) show that the transition shifts to lower weight STDs $\sigma$ for larger $K$, but apart from this fact the transitions are virtually identical for low and high $K$ values. Comparing

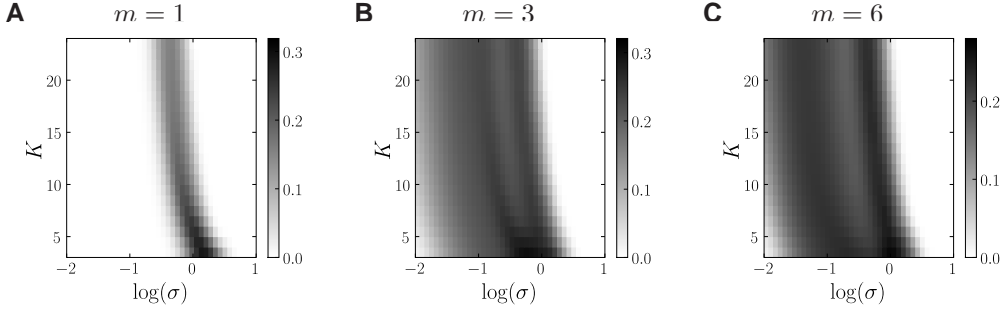

Figure 4: Mean-field predictor $p_\infty$ for computational power for different quantization levels $m$ as a function of the STD $\sigma$ of the weights and in-degree $K$. **A)** $m = 1$. **B)** $m = 3$. **C)** $m = 6$. Compare this result to the numerically determined performance $p_{\text{exp}}$ plotted in Fig. 1.

Fig. 3D with Fig. 1C, one sees that the rank measure does not accurately predict the whole region of good performance for high resolution reservoirs. It also does not predict the observed bifurcation in the zones of optimal performance, a phenomenon that is reproduced by the mean-field predictor introduced in the following section.

## 4 Mean-Field Predictor for Computational Performance

The question why and to what degree certain non-autonomous dynamical systems are useful devices for online computations has been addressed theoretically amongst others in [10]. There, the computational performance of networks of randomly connected threshold gates was linked to their separation property (for a formal definition see [2]): It was shown that only networks which exhibit sufficiently different network states for different instances of the input stream, i.e. networks that separate the input, can compute complex functions of the input stream. Furthermore, the authors introduced an accurate predictor for the computational capabilities for the considered type of networks based on the separation capability which was quantified via a simple mean-field approximation of the Hamming distance between different network states.

Here we aim at extending this approach to a larger class of networks, the class of quantized ESNs introduced above. However a severe problem arises when directly applying the mean-field theory developed in [10] to quantized ESNs with a quantization level $m > 1$: Calculation of the important quantities becomes computationally infeasible as the state space of a network grows exponentially with $m$. Therefore we introduce a modified mean-field predictor which can be efficiently computed and which still has all desirable properties of the one introduced in [10].

Suppose the target output of the network at time $t$ is a function $f_T \in F = \{f | f : \{-1, 1\}^n \to \{-1, 1\}\}$ of the $n$ bits $u(t - \tau - 1), \ldots, u(t - \tau - n)$ of the input stream $u(\cdot)$ with delay $\tau$ as described in Sec. 2. In order to exhibit good performance on an arbitrary $f_T \in F$, pairs of inputs that differ in at least one of the $n$ bits have to be mapped by the network to different states at time $t$. Only then, the linear classifier is able to assign the inputs to different function values. In order to quantify this so-called separation property of a given network, we introduce the normalized distance $d(k)$: It measures the average distance between two networks states $\mathbf{x}^1(t) = (x_1^1(t), \ldots, x_N^1(t))$ and $\mathbf{x}^2(t) = (x_1^2(t), \ldots, x_N^2(t))$ arising from applying to the same network two input streams $u^1(\cdot)$ and $u^2(\cdot)$ which only differ in the single bit at time $t - k$, i.e. $u^2(t - k) = -u^1(t - k)$. Formally we define[7]:

$$d(k) = \frac{1}{N} \left\langle \left\| \mathbf{x}^1(t) - \mathbf{x}^2(t) \right\|_1 \right\rangle.$$

The average $\langle . \rangle$ is taken over all inputs $u^1(\cdot)$, $u^2(\cdot)$ from the ensemble defined above, all initial conditions of the network and all circuits $C$. However, a good separation of the $n$ bits, i.e. $d(k) \gg 0$, $\tau < k \leq n + \tau$, is a necessary but not a sufficient condition for the ability of the network to calculate the target function. Beyond this, it is desired that the network "forgets" all (for the

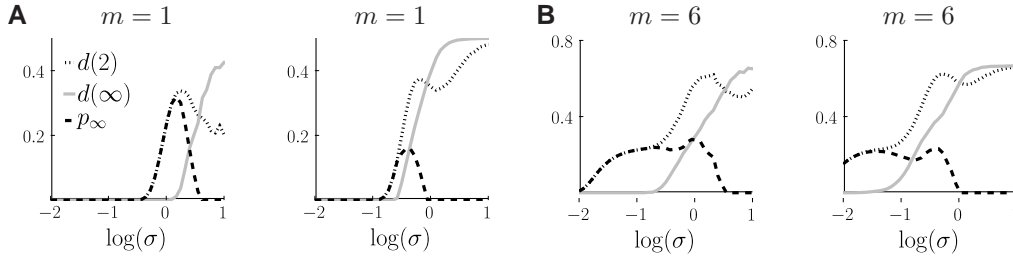

Figure 5: Contributions $d(2)$ (dotted) and $d(\infty)$ (solid gray) to the mean-field predictor $p_\infty$ (dashed line) for different quantization levels $m \in \{1, 6\}$ and different in-degrees $K \in \{3, 24\}$ as a function of STD $\sigma$ of the weights. The plots show slices of the 2d plots Fig. 4A and C for constant $K$. **A**) For $m = 1$ it can be seen that the region in $\log(\sigma)$-space with high $d(2)$ and low $d(\infty)$ is significantly larger for $K = 3$ than for $K = 24$. **B**) For $m = 6$ this region is roughly independent of K except a shift.

target function) irrelevant bits $u(t - k)$, $k > n + \tau$ of the input sufficiently fast, i.e. $d(k) \approx 0$ for $k > n + \tau$. We use the limit $d(\infty) = \lim_{k \to \infty} d(k)$ to quantify this irrelevant separation which signifies sensitivity to initial conditions (making the reservoir not time invariant). Hence, we propose the quantity $p_\infty$ as a heuristic predictor for computational power:

$$p_\infty = \max\{d(2) - d(\infty), 0\}.$$

As the first contribution to $p_\infty$ we chose $d(2)$ as it reflects the ability of a network to perform a combination of two mechanisms: In order to exhibit a high value for $d(2)$ the network has to separate the inputs at the time step $t - 2$ and to sustain the resulting state distance via its recurrent dynamics in the next time step $t - 1$. We therefore consider $d(2)$ to be a measure for input separation on short time-scales relevant for the target function. $p_\infty$ is calculated using a mean-field model similar to the one presented in [10] which itself is rooted in the annealed approximation (AA) introduced in [13]. In the AA one assumes that the circuit connectivity and the corresponding weights are drawn iid. at every time step. Although being a drastic simplification, the AA has been shown to yield good results in the large system size limit $N \to \infty$. The main advantage of $p_\infty$ over the the predictor defined in [10] (the NM-separation) is that the calculation of $p_\infty$ only involves taking the average over one input stream (as the $u^2(\cdot)$ is a function of $u^1(\cdot)$) compared to taking the average over two independent inputs needed for the NM-separation, resulting in a significantly reduced computation time.

In Fig. 4 the predictor $p_\infty$ is plotted as a function of the STD $\sigma$ of the weight distribution and the in-degree $K$ for three different values of the quantization level $m \in \{1, 3, 6\}$. When comparing these results with the actual network performance $p_{\exp}(\text{PAR})$ on the PAR-task plotted in Fig. 1 one can see that $p_\infty$ serves as a reliable predictor for $p_{\exp}$ of a network for sufficiently small $m$. For larger values of $m$ the predictor $p_\infty$ starts to deviate from the true performance. The dominant effect of the quantization level $m$ on the performance discussed in Sec. 2 is well reproduced by $p_\infty$: For $m = 1$ the in-degree $K$ has a considerable impact, i.e. for large $K$ maximum performance drops significantly. For $m > 2$ however, for larger values of $K$ there also exists a region in the parameter space exhibiting maximum performance.

The interplay between the two contributions $d(2)$ and $d(\infty)$ of $p_\infty$ delivers insight into the dependence of $p_{\exp}$ on the network parameters. A high value of $d(2)$ corresponds to a good separation of inputs on short time scales relevant for the target task, a property that is found predominantly in networks that are not strongly input driven. A small value of $d(\infty)$ guarantees that inputs on which the target function assumes the same value are mapped to nearby network states and thus a linear readout is able to assign them to the same class irrespectively of their irrelevant remote history. For $m = 1$, as can be seen in Fig. 5 the region in $\log(\sigma)$ space where both conditions for good performance are present decreases for growing $K$. In contrast, for $m > 2$ a reverse effect is observed: for increasing $K$ the parameter range for $\sigma$ fulfilling the two opposing conditions for good performance grows moderately resulting in a large region of high $p_\infty$ for high in-degree $K$. This observation is in close analogy to the behavior of the rank measure discussed in Sec. 3. Also note that $p_\infty$ predicts the novel bifurcation effect also observed in Fig. 1.

# 5 Discussion

By interpolating between the ESN and LSM approaches to RC, this work provides new insights into the question of what properties of a dynamical system lead to improved computational performance: Performance is optimal at the order-chaos phase transition, and the broader this transition regime, the better will the performance of the system be. We have confirmed this hypothesis by several analyses, including a new theoretical mean-field predictor that can be computed very efficiently.The importance of a gradual order-chaos phase transition could explain why ESNs are more often used for applications than LSMs. Although they can have very similar performance on a given task [7], it is significantly harder to create a LSM which operates at the edge-of-chaos: the excitation and inhibition in the network need to be finely balanced because there tends to be a very abrupt transition from an ordered to a epileptic state. For ESNs however, there is a broad parameter range in which they perform well. It should be noted that the effect of quantization cannot just be emulated by additive or multiplicative iid. or correlated Gaussian noise on the output of analog neurons. The noise degrades performance homogeneously and the differences in the influence of the in-degree observed for varying quantization levels cannot be reproduced. The finding that binary reservoirs have superior performance for low in-degree stands in stark contrast to the fact that cortical neurons have very high in-degrees of over $10^4$. This raises the interesting question which properties and mechanisms of cortical circuits not accounted for in this article contribute to their computational power. In view of the results presented in this article, such mechanisms should tend to soften the phase transition between order and chaos.

### Acknowledgments

Written under partial support by the FWO Flanders project # G.0088.09, the Photonics@be Interuniversity Attraction Poles program (IAP 6/10), the Austrian Science Fund FWF projects # P17229-N04, # S9102-N13 and projects # FP6-015879 (FACETS), # FP7-216593 (SECO) of the EU.

## Footnotes

[1]Shown by results of unpublished experiments which have also been reported by the lab of Jaeger through personal communication.

[2] $\kappa$ is defined as $(c - c_l)/(1 - c_l)$ where $c$ is the fraction of correct trials and $c_l$ is the chance level. The sum in eq. (1) was truncated at $\tau = 8$, as the performance was negligible for higher delays $\tau > 8$ for the network size $N = 150$.

[3] All logarithms are taken to the basis 10, i.e. $\log = \log_{10}$ if not stated otherwise.

[4] The Lyapunov coefficient $\lambda$ was determined in the following way. After 20 initial simulation steps the smallest admissible (for $m$) state difference $\delta_0(m) = 2^{1-m}$ was introduced in a single network unit and the resulting state difference $\delta$ after one time step was measured averaged over $10^5$ trials with randomly generated networks, initial states and input streams. The initial states of all neurons were iid. uniformly over $\mathcal{S}_m$. $\lambda$ was then determined by $\lambda = \ln(\delta/\delta_0(m))$.

[5]The initial states of all neurons were iid. uniformly over $\mathcal{S}_m$. The rank of the matrix was estimated by singular value decomposition on the network states after 15 time steps of simulation.

[6]First, we drew each of the last three bits $\tilde{u}(13), \ldots, \tilde{u}(15)$ independently from a uniform distribution over $\{-1, 1\}$. For each input stream $\tilde{u}_i(1), \ldots, \tilde{u}_i(15)$ we drew $\tilde{u}_i(1), \ldots, \tilde{u}_i(12)$ independently from a uniform distribution over $\{-1, 1\}$ and set $\tilde{u}_i(t) = \tilde{u}(t)$ for $t = 13, \ldots, 15$.

[7]For vectors $\mathbf{x} = (x_1, x_2, \ldots) \in \mathbb{R}^N$ we use the Manhattan norm $\|\mathbf{x}\|_1 := \sum_{i=1}^N |x_i|$

# References

[1] H. Jaeger. The "echo state" approach to analyzing and training recurrent neural networks. GMD Report 148, German National Research Center for Information Technology, 2001.

[2] W. Maass, T. Natschläger, and H. Markram. Real-time computing without stable states: A new framework for neural computation based on perturbations. *Neural Computation*, 14(11):2531–2560, 2002.

[3] Kristof Vandoorne, Wouter Dierckx, Benjamin Schrauwen, David Verstraeten, Roel Baets, Peter Bienstman, and Jan Van Campenhout. Toward optical signal processing using photonic reservoir computing. *Optics Express*, 16(15):11182–11192, 8 2008.

[4] H. Jaeger and H. Haas. Harnessing nonlinearity: predicting chaotic systems and saving energy in wireless communication. *Science*, 304:78–80, 2004.

[5] D. Verstraeten, B. Schrauwen, D. Stroobandt, and J. Van Campenhout. Isolated word recognition with the liquid state machine: a case study. *Information Processing Letters*, 95(6):521–528, 2005.

[6] P. Joshi and W. Maass. Movement generation with circuits of spiking neurons. *Neural Computation*, 17(8):1715–1738, 2005.

[7] D. Verstraeten, B. Schrauwen, M. D'Haene, and D. Stroobandt. A unifying comparison of Reservoir Computing methods. *Neural Networks*, 20:391–403, 2007.

[8] H. Jaeger. Echo state networks. *Scholarpedia*, 2(9):2330, 2007.

[9] S. Häusler and W. Maass. A statistical analysis of information processing properties of lamina-specific cortical microcircuit models. *Cerebral Cortex*, 17(1):149–162, 2007.

[10] N. Bertschinger and T. Natschläger. Real-time computation at the edge of chaos in recurrent neural networks. *Neural Computation*, 16(7):1413–1436, 2004.

[11] R. Legenstein and W. Maass. Edge of chaos and prediction of computational performance for neural microcircuit models. *Neural Networks*, pages 323–334, 2007.

[12] R. Legenstein and W. Maass. What makes a dynamical system computationally powerful? In S. Haykin, J. C. Principe, T.J. Sejnowski, and J.G. McWhirter, editors, *New Directions in Statistical Signal Processing: From Systems to Brain*, pages 127–154. MIT Press, 2007.

[13] B. Derrida and Pomeau Y. Random networks of automata: A simple annealed approximation. *Europhysics Letters*, 1(2):45–49, 1986.

